# Encoding Labeled Graphs by Labeling RAAM

**Alessandro  Sperduti**[*]
Department of Computer Science
Pisa University
Corso Italia 40, 56125 Pisa, Italy

## Abstract

In this paper we propose an extension to the RAAM by Pollack. This extension, the Labeling RAAM (LRAAM), can encode labeled graphs with cycles by representing pointers explicitly. Data encoded in an LRAAM can be accessed by pointer as well as by content. Direct access by content can be achieved by transforming the encoder network of the LRAAM into an analog Hopfield network with hidden units. Different access procedures can be defined depending on the access key. Sufficient conditions on the asymptotical stability of the associated Hopfield network are briefly introduced.

## 1  INTRODUCTION

In the last few years, several researchers have tried to demonstrate how symbolic structures such as lists, trees, and stacks can be represented and manipulated in a connectionist system, while still preserving all the computational characteristics of connectionism (and extending them to the symbolic representations) (Hinton, 1990; Plate, 1991; Pollack, 1990; Smolensky, 1990; Touretzky, 1990). The goal is to highlight the potential of the connectionist approach in handling domains of structured tasks. The common background of their ideas is an attempt to achieve distal access and consequently compositionality. The RAAM model, proposed by Pollack (Pollack, 1990), is one example of how a neural network can discover compact recursive

---

[*]Work partially done while at the International Computer Science Institute, Berkeley.

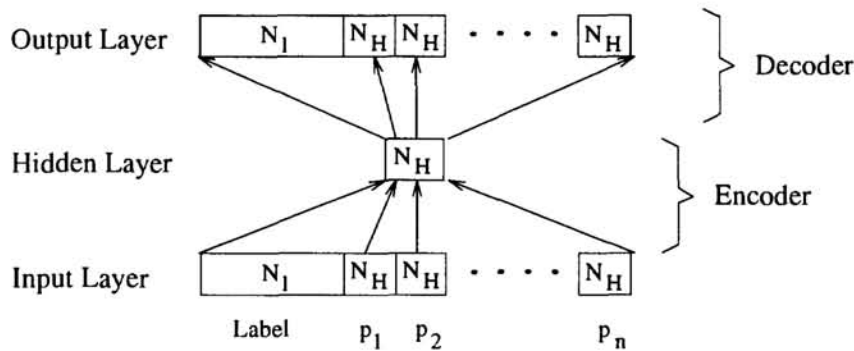

Figure 1: The network for a general LRAAM. The first layer of the network implements an encoder; the second layer, the corresponding decoder.

distributed representations of trees with a fixed branching factor.

This paper presents an extension of the RAAM, the Labeling RAAM (LRAAM). An LRAAM allows one to store a label for each component of the structure to be represented, so as to generate reduced representations of labeled graphs. Moreover, data encoded in an LRAAM can be accessed not only by pointer but also by content. In Section 2 we present the network and we discuss some technical aspects of the model. The possibility to access data by content is presented in Section 3. Some stability results are introduced in Section 4, and the paper is closed by discussion and conclusions in Section 5.

## 2   THE NETWORK

The general structure of the network for an LRAAM is shown in Figure 1. The network is trained by backpropagation to learn the identity function. The idea is to obtain a compressed representation (hidden layer activation) of a node of a labeled graph by allocating a part of the input (output) of the network to represent the label ($N_l$ units) and the remaining part to represent one or more pointers. This representation is then used as pointer to the node. To allow the recursive use of these compressed representations, the part of the input (output) layer which represents a pointer must be of the same dimension as the hidden layer ($N_H$ units). Thus, a general LRAAM is implemented by a $N_I - N_H - N_I$ feed-forward network, where $N_I = N_l + nN_H$, and $n$ is the number of pointer fields.

Labeled graphs can be easily encoded using an LRAAM. Each node of the graph only needs to be represented as a record, with one field for the label and one field for each pointer to a connected node. The pointers only need to be logical pointers, since their actual values will be the patterns of hidden activation of the network. At the beginning of learning, their values are set at random. A graph is represented by a list of these records, and this list constitutes the initial training set for the LRAAM. During training the representations of the pointers are consistently updated according to the hidden activations. Consequently, the training set is dynamic. For example, the network for the graph shown in Figure 2 can be trained as follows:

| input | | hidden | | output |
|---|---|---|---|---|
| $(L_1\ d_{n2}\ d_{n4}\ d_{n5})$ | $\rightarrow$ | $d'_{n1}$ | $\rightarrow$ | $(L''_1\ d''_{n2}\ d''_{n4}\ d''_{n5})$ |
| $(L_2\ d_{n3}\ d_{n4}\ nil)$ | $\rightarrow$ | $d'_{n2}$ | $\rightarrow$ | $(L''_2\ d''_{n3}\ d''_{n4}\ nil'')$ |
| $(L_3\ d_{n6}\ nil\ nil)$ | $\rightarrow$ | $d'_{n3}$ | $\rightarrow$ | $(L''_3\ d''_{n6}\ nil''\ nil'')$ |
| $(L_4\ d_{n6}\ d_{n3}\ nil)$ | $\rightarrow$ | $d'_{n4}$ | $\rightarrow$ | $(L''_4\ d''_{n6}\ d''_{n3}\ nil'')$ |
| $(L_5\ d_{n4}\ d_{n6}\ nil)$ | $\rightarrow$ | $d'_{n5}$ | $\rightarrow$ | $(L''_5\ d''_{n4}\ d''_{n6}\ nil'')$ |
| $(L_6\ nil\ nil\ nil)$ | $\rightarrow$ | $d'_{n6}$ | $\rightarrow$ | $(L''_6\ nil''\ nil''\ nil'')$ |

where $L_i$ and $d_{ni}$ are respectively the label and the pointer (reduced descriptor to the i-th node. For the sake of simplicity, the void pointer is represented by a single symbol, *nil*, but each instance of it must be considered as being different. This statement will be made clear in the next section.

Once the training is complete, the patterns of activation representing pointers can be used to retrieve information. Thus, for example, if the activity of the hidden units of the network is clamped to $d_{n1}$, the output of the network becomes $(L_1, d_{n2}, d_{n4}, d_{n5})$, enabling further retrieval of information by decoding $d_{n2}$, or $d_{n4}$, or $d_{n5}$, and so on. Note that more labeled graphs can be encoded in the same LRAAM.

## 2.1 THE VOID POINTER PROBLEM

In the RAAM model there is a termination problem in the decoding of a compressed representation: due to approximation errors introduced during decoding, it is not clear when a decoded pattern is a terminal or a nonterminal. One solution is to test for "binary-ness", which consists in checking whether all the values of a pattern are above $1 - \tau$ or below $\tau$, $\tau > 0$, $\tau \ll 1$. However, a nonterminal may also pass the test for "binary-ness".

One advantage of LRAAM over RAAM is the possibility to solve the problem by allocating one bit of the label for each pointer to represent if the pointer is void or not. This works better than fixing a particular pattern for the void pointer, such as a pattern with all the bits to 1 or 0 or -1 (if symmetrical sigmoids are used). Simulations performed with symmetrical sigmoids showed that the configurations with all bits equal to 1 or -1 were also used by non void pointers, whereas the configuration with all bits set to zero considerably reduced the rate of convergence. Using a part of the label to solve the problem is particularly efficient, since the pointer fields are free to take on any configuration when they are void, and this increases the freedom of the system. To facilitate learning, the output activation of the void pointers in one epoch is used as an input activation in the next epoch. Experimentation showed fast convergence to different fixed points for different void

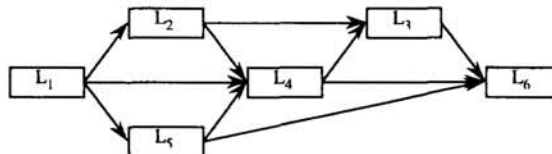

Figure 2: An example of a labeled graph.

pointers. For this reason, we claimed that each occurrence of the void pointer is different, and that the *nil* symbol can be considered as a "don't care" symbol.

## 2.2   REPRESENTATION OF THE TRAINING SET

An important question about the way a graph is represented in the training set is which aspects of the representation itself can make the encoding task harder or easier. In (Sperduti, 1993a) we made a theoretical analysis on the constraints imposed by the representation on the set of weights of the LRAAM, under the hypotheses of perfect learning (zero total error after learning) and linear output units. Our findings were:

*i)* pointers to nodes belonging to the same cycle of length $k$ and represented in the same pointer field $p$, must be eigenvectors of the matrix $(W^{(p)})^k$, where $W^{(p)}$ is the connection matrix between the hidden layer and the output units representing the pointer field $p$;

*ii) confluent pointers*, i.e., pointers to the same node represented in the same pointer field $p$ (of different nodes), contribute to reducing the rank of the matrix $W^{(p)}$, the actual rank is however dependent on the constraints imposed by the label field and the other pointer fields.

We have observed that different representations of the same structure can lead to very different learning performances. However, representations with roughly the same number of non void pointers for each pointer field, with cycles represented in different pointer fields and with confluent pointers seem to be more effective.

## 3   ACCESS BY CONTENT

Retrieval of coded information is performed in RAAM through the pointers. All the terminals and nonterminals can be retrieved recursively by the pointers to the whole tree encoded in a RAAM. If direct access to a component of the tree is required, the pointer to the component must be stored and used on demand.

Data encoded in an LRAAM can also be accessed directly by content. In fact, an LRAAM network can be transformed into an analog Hopfield network with one hidden layer and asymmetric connection matrix by feeding back its output into its input units. [1] Because each pattern is structured in different fields, different access

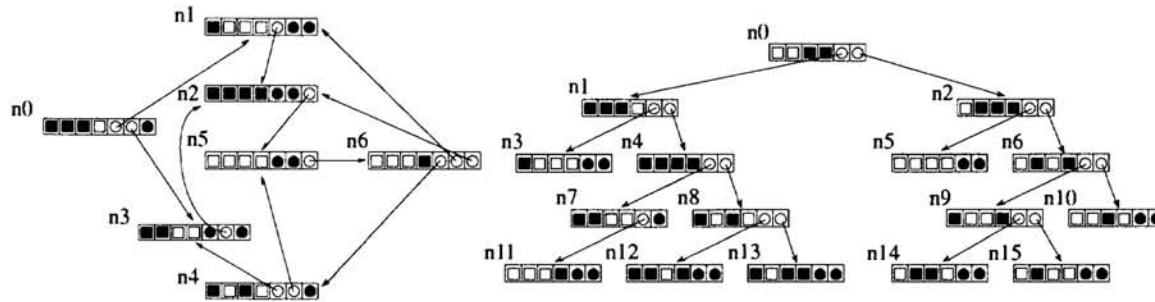

Figure 3: The labeled graph encoded in a 16-3-16 LRAAM (5450 epochs), and the labeled tree encoded in a 18-6-18 LRAAM (1719 epochs).

procedures can be defined on the Hopfield network according to the type of access key. An access procedure is defined by:

1. choosing one or more fields in the input layer according to the access key(s);
2. clamping the output of such units to the the access key(s);
3. setting randomly the output of the remaining units in the network;
4. letting the remaining units of the network to relax into a stable state.

A validation test of the reached stable state can be performed by:

1. unfreezing the clamped units in the input layer;
2. if the stable state is no longer stable the result of the procedure is considered *wrong* and another run is performed;
3. otherwise the stable state is considered a *success*.

This validation test, however, sometimes can fail to detect an erroneous retrieval (*error*) because of the existence of spurious stable states that share the same known information with the desired one.

The results obtained by the access procedures on an LRAAM codifying the graph and on an LRAAM codifying the tree shown in Figure 3 are reported in Table 1. For each procedure 100 trials were performed. The "mean" column in the table reports the mean number of iterations employed by the Hopfield network to converge. The access procedure by outgoing pointers was applied only for the tree. It can be seen from Table 1 that the performances of the access procedures were high for the graph (no errors and no wrong retrievals), but not so good for the tree, in particular for the access by label procedure, due to spurious memories. It is interesting to note that the access by label procedure is very efficient for the leaves of the tree. This feature can be used to build a system with two identical networks, one accessed by pointer and the other by content. The search for a label proceeds simultaneously into the two networks. The network accessed by pointer will be very fast to respond when the label is located on a node at lower levels of the tree, and the network accessed by content will be able to respond correctly and very fast [2] when the label is located on a node at higher levels of the tree.

| GRAPH: Access by Label | | | | TREE: Access by Label | | | | |
|---|---|---|---|---|---|---|---|---|
| key(s) | success | wrong | error | mean | key | success | wrong | error | mean |
| $l_0$ | 100% | 0% | 0% | 7.35 | $l_0$ | 0% | 100% | 0% | 16.48 |
| $l_1$ | 100% | 0% | 0% | 36.05 | $l_1$ | 94% | 6% | 0% | 14.57 |
| $l_2$ | 100% | 0% | 0% | 6.04 | $l_2$ | 47% | 53% | 0% | 16.92 |
| $l_3$ | 100% | 0% | 0% | 3.99 | $l_3$ | 100% | 0% | 0% | 18.07 |
| $l_4$ | 100% | 0% | 0% | 23.12 | $l_4$ | 97% | 0% | 3% | 32.64 |
| $l_5$ | 100% | 0% | 0% | 18.12 | $l_5$ | 100% | 0% | 0% | 16.03 |
| $l_6$ | 100% | 0% | 0% | 29.26 | $l_6$ | 49% | 51% | 0% | 27.50 |
| TREE: Access by Children Pointers | | | | | $l_7$ | 42% | 58% | 0% | 27.10 |
| $(d_1, d_2)$ | 49% | 51% | 0% | 6.29 | $l_8$ | 57% | 43% | 0% | 62.45 |
| $(d_3, d_4)$ | 10% | 90% | 0% | 8.55 | $l_9$ | 20% | 0% | 80% | 14.75 |
| $(d_5, d_6)$ | 40% | 60% | 0% | 12.48 | $l_{10}$ | 100% | 0% | 0% | 19.11 |
| $(d_7, d_8)$ | 78% | 22% | 0% | 6.57 | $l_{11}$ | 100% | 0% | 0% | 10.83 |
| $(d_9, d_{10})$ | 9% | 91% | 0% | 6.22 | $l_{12}$ | 100% | 0% | 0% | 19.12 |
| $d_{11}^{(*)}$ | 14% | 86% | 0% | 14.01 | $l_{13}$ | 29% | 71% | 0% | 23.87 |
| $(d_{12}, d_{13})$ | 14% | 86% | 0% | 7.87 | $l_{14}$ | 100% | 0% | 0% | 12.09 |
| $(d_{14}, d_{15})$ | 28% | 72% | 0% | 6.07 | $l_{15}$ | 100% | 0% | 0% | 13.11 |
| (*) one pointer | | | | | | | | | |

Table 1: Results obtained by the access procedures.

# 4    STABILITY RESULTS

In the LRAAM model two stability problems are encountered. The first one arises when considering the decoding of a pointer along a cycle of the encoded structures. Since the decoding process suffers, in general, of approximation errors, it may happen that the decoding diverges from the correct representations of the pointers belonging to the cycle. Thus, it is fundamental to discover under which conditions the representations obtained for the pointers are asymptotically stable with respect to the pointer transformation. In fact, if the representations are asymptotically stable, the errors introduced by the decoding function are automatically corrected. The following theorem can be proven (Sperduti, 1993b):

**Theorem 1** *A decoding sequence*

$$\vec{d}^{(i_{j}+1)} = \boldsymbol{F}^{(p_{i_j})}(\vec{d}^{(i_j)}), \quad j = 0, \ldots, L \tag{1}$$

*with $\vec{d}^{(i_{L+1})} = \vec{d}^{(i_0)}$, satisfying*

$$\sum_{k=1}^{m} |b_{ik}| < 1, \quad i = 1, \ldots, m \tag{2}$$

*for some index $p_{i_q}$, $q = 0, \ldots, L$, is asymptotically stable, where $b_{ik}$ is the $(i, k)$th element of a matrix $\boldsymbol{B}$, given by*

$$\boldsymbol{B} = \boldsymbol{J}^{(p_{i_q})}(\vec{d}^{(i_q)})\boldsymbol{J}^{(p_{i_q-1})}(\vec{d}^{(i_q-1)}) \cdots \boldsymbol{J}^{(p_{i_0})}(\vec{d}^{(i_0)})\boldsymbol{J}^{(p_{i_L})}(\vec{d}^{(i_L)}) \cdots \boldsymbol{J}^{(p_{i_q+1})}(\vec{d}^{(i_q+1)}).$$

In the statement of the theorem, $\boldsymbol{F}^{(p_j)}(\vec{d}) = \boldsymbol{F}(\boldsymbol{D}^{(p_j)}\vec{d} + \vec{\theta}_O^{(p_j)})$ is the transformation of the reduced descriptor (pointer) $\vec{d}$ by the pointer field $p_j$, and $\boldsymbol{J}^{(p_j)}(\vec{d})$ is its

Jacobian matrix. As a corollary of this theorem we have that *if at least one pointer belonging to the cycle has saturated components, then the cycle is asymptotically stable with respect to the decoding process.* Moreover, the theorem can be applied with a few modifications to the stability analysis of the fixed points of the associated Hopfield network.

The second stability problem consists into the discovering of sufficient conditions under which the property of asymptotical stability of a fixed point in one particular constrained version of the associated Hopfield network, i.e., an access procedure, can be extended to related fixed points of different constrained versions of it, i.e., access procedures with more information or different information. The result of Theorem 1 was used to derive three theorems regarding this issue (see (Sperduti, 1993b)).

## 5   DISCUSSION AND CONCLUSIONS

The LRAAM model can be seen from various perspectives. It can be considered as an extension of the RAAM model, which allows one to encode not only trees with information on the leaves, but also labeled graphs with cycles. On the other hand, it can be seen as an approximate method to build analog Hopfield networks with a hidden layer. An LRAAM is probably somewhere in between. In fact, although it extends the representational capabilities of the RAAM model, it doesn't possess the same synthetic capabilities as the RAAM, since it explicitly uses the concept of pointer. Different subsets of units are thus used to codify labels and pointers. In the RAAM model, using the same set of units to codify labels and reduced representations is a more natural way of integrating a previously developed reduced descriptor as a component of a new structure. In fact, this ability was Pollack's original rationale behind the RAAM model, since with this ability it is possible to fill a linguistic role with the reduced descriptor of a complex sentence. In the LRAAM model the same target can be reached, but less naturally. There are two possible solutions. One is to store the pointer of some complex sentence (or structure, in general), which was previously developed, in the label of a new structure. The other solution would be to have a particular label value which tells us that the information we are looking for can be retrieved using one conventional or particular pointer among the current ones.

An issue strictly correlated with this is that, even if in an LRAAM it is possible to encode a cycle, what we get from the LRAAM is not an explicit reduced representation of the cycle, but several pointers to the components of the cycle forged in such a way that the information on the cycle is only represented implicitly in each of them. However, the ability to synthesize reduced descriptors for structures with cycles is what makes the difference between the LRAAM and the RAAM. The only system that we know of which is able to represent labeled graphs is the DUAL system proposed by Dyer (Dyer, 1991). It is able to encode small labeled graphs representing relationships among entities. However, the DUAL system cannot be considered as being on the same level as the LRAAM, since it devises a reduced representation of a set of functions relating the components of the graph rather than a reduced representation for the graph. Potentially also Holographic Reduced Representations (Plate, 1991) are able to encode cyclic graphs.

The LRAAM model can also be seen as an extension of the Hopfield networks philosophy. A relevant aspect of the use of the Hopfield network associated with an LRAAM, is that the access procedures defined on it can efficiently exploit subsets of the weights. In fact, their use corresponds to generating several smaller networks from a large network, one for each kind of access procedure, each specialized on a particular feature of the stored data. Thus, by training a single network, we get several useful smaller networks.

In conclusion an LRAAM has several advantages over a standard RAAM. Firstly, it is more powerful, since it allows to encode directed graphs where each node has a bounded number of outgoing arcs. Secondly, an LRAAM allows direct access to the components of the encoded structure not only by pointer, but also by content. Concerning the applications where LRAAMs can be exploited, we believe there are at least three possibilities: in knowledge representation, by encoding *Conceptual Graphs* (Sowa, 1984); in unification, by representing *terms* in restricted domains (Knight, 1989); in image coding, by storing *Quadtrees* (Samet, 1984);

## Footnotes

[1] Experimental results have shown that there is a high correlation between elements of $W^{(h)}$ (the set of weights from the input to the hidden layer) and the corresponding elements in $W^{(o)^T}$ (the set of weights from the hidden to the output layer). This is particularly true for weights corresponding to units of the label field. Such result is not a total surprise, since in the case of a static training set, the error function of a linear encoder network has been proven to have a unique global minimum corresponding to the projection onto the subspace generated by the first principal vectors of a covariance matrix associated with the training set (Baldi & Hornik, 1989). This implies that the weights matrices are transposes of each other unless there is an invertible transformation between them (see also (Bourlard & Kamp, 1988)).

[2] Assuming an analog implementation of the Hopfield network.

## References

P. Baldi & K. Hornik. (1989) Neural networks and principal component analysis: Learning from examples without local minima. *Neural Networks*, 2:53-58.

H. Bourlard & Y. Kamp. (1988) Auto-association by multilayer perceptrons and singular value decomposition. *Biological Cybernetics*, 59:291-294.

M. G. Dyer. (1991) *Symbolic NeuroEngineering for Natural Language Processing: A Multi-level Research Approach.*, volume 1 of *Advances in Connectionist and Neural Computation Theory*, pages 32-86. Ablex.

G. E. Hinton. (1990) Mapping part-whole hierarchies into connectionist networks. *Artificial Intelligence*, 46:47-75.

K. Knight. (1989) Unification: A multidisciplinary survey. *ACM Computing Surveys*, 21:93-124.

T. Plate. (1991) Holographic reduced representations. Technical Report CRG-TR-91-1, Department of Computer Science, University of Toronto.

J. B. Pollack. (1990) Recursive distributed representations. *Artificial Intelligence*, 46(1-2):77-106.

H. Samet. (1984) The quadtree and related hierarchical data structures. *ACM Computing Surveys*, 16:187-260.

P. Smolensky. (1990) Tensor product variable binding and the representation of symbolic structures in connectionist systems. *Artificial Intelligence*, 46:159-216.

J.F. Sowa. (1984) *Conceptual Structures: Information Processing in Mind and Machine.* Addison-Wesley.

A. Sperduti. (1993a) Labeling RAAM. TR 93-029, ICSI, Berkeley.

A. Sperduti. (1993b) On some stability properties of the LRAAM model. TR 93-031, ICSI, Berkeley.

D. S. Touretzky. (1990) Boltzcons: Dynamic symbol structures in a connectionist network. *Artificial Intelligence*, 46:5-46.


